# Gated Softmax Classification

**Roland Memisevic**
Department of Computer Science
ETH Zurich
Switzerland
roland.memisevic@gmail.com

**Christopher Zach**
Department of Computer Science
ETH Zurich
Switzerland
chzach@inf.ethz.ch

**Geoffrey Hinton**
Department of Computer Science
University of Toronto
Canada
hinton@cs.toronto.edu

**Marc Pollefeys**
Department of Computer Science
ETH Zurich
Switzerland
marc.pollefeys@inf.ethz.ch

## Abstract

We describe a "log-bilinear" model that computes class probabilities by combining an input vector multiplicatively with a vector of binary latent variables. Even though the latent variables can take on exponentially many possible combinations of values, we can efficiently compute the exact probability of each class by marginalizing over the latent variables. This makes it possible to get the exact gradient of the log likelihood. The bilinear score-functions are defined using a three-dimensional weight tensor, and we show that factorizing this tensor allows the model to encode invariances inherent in a task by learning a dictionary of invariant basis functions. Experiments on a set of benchmark problems show that this fully probabilistic model can achieve classification performance that is competitive with (kernel) SVMs, backpropagation, and deep belief nets.

## 1 Introduction

Consider the problem of recognizing an image that contains a single hand-written digit that has been approximately normalized but may have been written in one of a number of different styles. Features extracted from the image often provide much better evidence for a *combination* of a class and a style than they do for the class alone. For example, a diagonal stroke might be highly compatible with an italic 1 or a non-italic 7. A short piece of horizontal stroke at the top right may be compatible with a very italic 3 or a 5 with a disconnected top. A fat piece of vertical stroke at the bottom of the image near the center may be compatible with a 1 written with a very thick pen or a narrow 8 written with a moderately thick pen so that the bottom loop has merged. If each training image was labeled with both the class and the values of a set of binary style features, it would make sense to use the image features to create a bipartite conditional random field (CRF) which gave low energy to combinations of a class label and a style feature that were compatible with the image feature. This would force the way in which local features were interpreted to be globally consistent about style features such as stroke thickness or "italicness". But what if the values of the style features are missing from the training data?

We describe a way of learning a large set of binary style features from training data that are only labeled with the class. Our "gated softmax" model allows the $2^K$ possible combinations of the $K$ learned style features to be integrated out. This makes it easy to compute the posterior probability of a class label on test data and easy to get the exact gradient of the log probability of the correct label on training data.

## 1.1 Related work

The model is related to several models known in the literature, that we discuss in the following. [1] describes a bilinear sparse coding model that, similar to our model, can be trained discriminatively to predict classes. Unlike in our case, there is no interpretation as a probabilistic model, and – consequently – not a simple learning rule. Furthermore, the model parameters, unlike in our case, are not factorized, and as a result the model cannot extract features which are shared among classes. Feature sharing, as we shall show, greatly improves classification performance as it allows for learning of invariant representations of the input.

Our model is similar to the top layer of the deep network discussed in [2], again, without factorization and feature sharing. We also derive and utilize discriminative gradients that allow for efficient training. Our model can be viewed also as a "degenerate" special case of the image transformation model described in [3], which replaces the output-image in that model with a "one-hot" encoded class label. The intractable objective function of that model, as a result, collapses into a tractable form, making it possible to perform exact inference.

We describe the basic model, how it relates to logistic regression, and how to perform learning and inference in the following section. We show results on benchmark classification tasks in Section 3 and discuss possible extensions in Section 4.

## 2 The Gated Softmax Model

### 2.1 Log-linear models

We consider the standard classification task of mapping an input vector $\boldsymbol{x} \in I\!\!R^n$ to a class-label $y$. One of the most common, and certainly oldest, approaches to solving this task is *logistic regression*, which is based on a log-linear relationship between inputs and labels (see, for example, [4]). In particular, using a set of *linear*, class-specific score functions

$$s_y(\boldsymbol{x}) = \boldsymbol{w}_y^{\mathrm{t}} \boldsymbol{x} \tag{1}$$

we can obtain probabilities over classes by exponentiating and normalizing:

$$p(y|\boldsymbol{x}) = \frac{\exp(\boldsymbol{w}_y^{\mathrm{t}} \boldsymbol{x})}{\sum_{y'} \exp(\boldsymbol{w}_{y'}^{\mathrm{t}} \boldsymbol{x})} \tag{2}$$

Classification decisons for test-cases $\boldsymbol{x}^{\mathrm{test}}$ are given by $\arg\max p(y|\boldsymbol{x}^{\mathrm{test}})$. Training amounts to adapting the vectors $\boldsymbol{w}_y$ by maximizing the average conditional log-probability $\frac{1}{N}\sum_\alpha \log p(y^\alpha|\boldsymbol{x}^\alpha)$ for a set $\{(\boldsymbol{x}^\alpha, y^\alpha)\}_{\alpha=1}^N$ of training cases. Since there is no closed form solution, training is typically performed using some form of gradient based optimization. In the case of two or more labels, logistic regression is also referred to as the "multinomial logit model" or the "maximum entropy model" [5]. It is possible to include additive "bias" terms $b_y$ in the definition of the score function (Eq. 1) so that class-scores are affine, rather than linear, functions of the input. Alternatively, we can think of the inputs as being in a "homogeneous" representation with an extra constant 1-dimension, in which biases are implemented implicitly.

Important properties of logistic regression are that (a) the training objective is convex, so there are no local optima, and (b) the model is probabilistic, hence it comes with well-calibrated estimates of uncertainty in the classification decision (ref. Eq. 2) [4]. Property (a) is shared with, and property (b) a possible advantage over, margin-maximizing approaches, like support vector machines [4].

### 2.2 A *log-bilinear* model

Logistic regression makes the assumption that classes can be separated in the input space with hyperplanes (up to noise). A common way to relax this assumption is to replace the linear separation manifold, and thus, the score function (Eq. 1), with a non-linear one, such as a neural network [4]. Here, we take an entirely different, probabilistic approach. We take the stance that we do not *know* what form the separation manifold takes on, and instead introduce a set of probabilistic hidden variables which cooperate to model the decision surface jointly. To obtain classification decisions at test-time and for training the model, we then need to *marginalize* over these hidden variables.

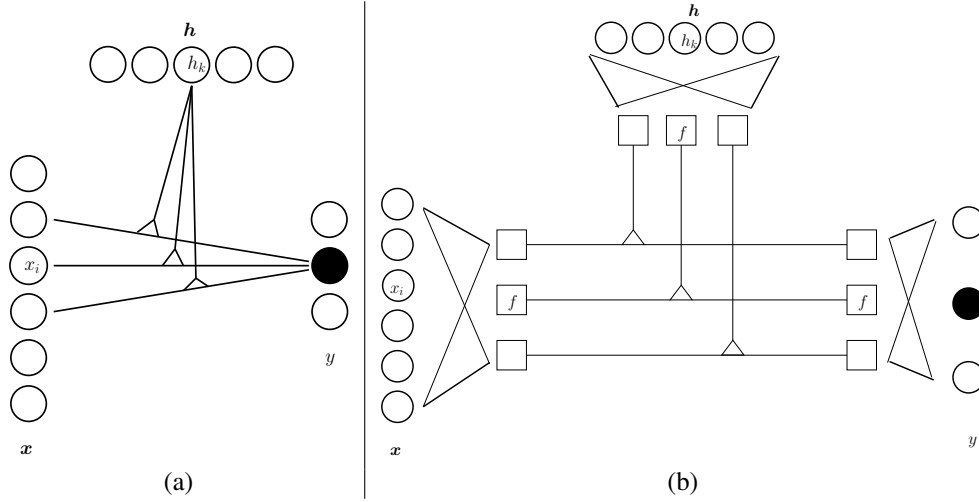

Figure 1: (a) A log-bilinear model: Binary hidden variables $h_k$ can *blend* in log-linear dependencies that connect input features $x_i$ with labels $y$. (b) Factorization allows for blending in a learned feature space.

More specifically, we consider the following variation of logistic regression: We introduce a vector $\boldsymbol{h}$ of *binary latent variables* $(h_1, \ldots, h_K)$ and replace the *linear* score (Eq. 1) with a *bilinear* score of $\boldsymbol{x}$ and $\boldsymbol{h}$:

$$s_y(\boldsymbol{x}, \boldsymbol{h}) = \boldsymbol{h}^{\mathrm{t}} W_y \boldsymbol{x}. \tag{3}$$

The bilinear score combines, quadratically, all pairs of input components $x_i$ with hidden variables $h_k$. The score for each class is thus a quadratic product, parameterized by a class-specific *matrix* $W_y$. This is in contrast to the inner product, parameterized by class-specific *vectors* $\boldsymbol{w}_y$, for logistic regression. To turn scores into probabilities we can again exponentiate and normalize

$$p(y, \boldsymbol{h} | \boldsymbol{x}) = \frac{\exp(\boldsymbol{h}^{\mathrm{t}} W_y \boldsymbol{x})}{\sum_{y' \boldsymbol{h}'} \exp(\boldsymbol{h}'^{\mathrm{t}} W_{y'} \boldsymbol{x})}. \tag{4}$$

In contrast to logistic regression, we obtain a distribution over both the hidden variables $\boldsymbol{h}$ *and* labels $y$. We get back the (input-dependent) distributions over labels with an additional *marginalization* over $\boldsymbol{h}$:

$$p(y | \boldsymbol{x}) = \sum_{\boldsymbol{h} \in \{0,1\}^K} p(y, \boldsymbol{h} | \boldsymbol{x}). \tag{5}$$

As with logistic regression, we thus get a distribution over labels $y$, conditioned on inputs $\boldsymbol{x}$. The parameters are the set of class-specific matrices $W_y$. As before, we can add bias terms to the score, or add a constant 1-dimension to $\boldsymbol{x}$ and $\boldsymbol{h}$. Note that for any *single and fixed* instantiation of $\boldsymbol{h}$ in Eq. 3, we obtain the logistic regression score (up to normalization), since the argument in the "$\exp()$" collapses to the class-specific row-vector $\boldsymbol{h}^t W_y$. Each of the $2^K$ summands in Eq. 5 is therefore exactly one logistic classifier, showing that the model is equivalent to a mixture of $2^K$ logistic regressors with shared weights. Because of the weight-sharing the number of parameters grows linearly not exponentially in the number of hidden variables. In the following, we let $W$ denote the three-way tensor of parameters (by "stacking" the matrices $W_y$).

The sum over $2^K$ terms in Eq. 5 seems to preclude any reasonably large value for $K$. However, similar to the models in [6], [7], [2], the marginalization can be performed in closed form and can be computed tractably by a simple re-arrangement of terms:

$$p(y | \boldsymbol{x}) = \sum_{\boldsymbol{h}} p(y, \boldsymbol{h} | \boldsymbol{x}) \propto \sum_{\boldsymbol{h}} \exp(\boldsymbol{h}^{\mathrm{t}} W_y \boldsymbol{x}) = \sum_{\boldsymbol{h}} \exp\left(\sum_{ik} W_{yik} x_i h_k\right) = \prod_k \left(1 + \exp\left(\sum_i W_{yik} x_i\right)\right) \tag{6}$$

This shows that the class probabilities decouple into a product of $K$ terms[1], each of which is a mixture of a uniform and an input-conditional "softmax". The model is thus a *product of experts* [8] (which is *conditioned* on input vectors $\boldsymbol{x}$). It can be viewed also as a "strange" kind of Gated Boltzmann Machine [9] that models a *single* discrete output variable $y$ using $K$ binary latent variables. As we shall show, it is the *conditioning* on the inputs $\boldsymbol{x}$ that renders this model useful.

Typically, training products of experts is performed using approximate, sampling based schemes, because of the lack of a closed form for the data probability [8]. The same is true for most *conditional* products of experts [9].

Note that in our case, the distribution that the model outputs is a distribution over a countable (and, in particular, fairly small[2]) number of possible values, so we can compute the constant $\Omega = \sum_{y'} \prod_k (1 + \exp(\sum_i W_{yik} x_i))$, that normalizes the left-hand side in Eqs. 6, efficiently. The same observation was utilized before in [6], [7], [10].

## 2.3 Sharing features among classes

The score (or "activation") that class label $y$ receives from each of the $2^K$ terms in Eq. 5 is a linear function of the inputs. A different class $y'$ receives activations from a *different*, non-overlapping set of functions. The number of parameters is thus: (number of inputs) $\times$ (number of labels) $\times$ (number of hidden variables). As we shall show in Section 3 the model can achieve fairly good classification performance.

A much more natural way to define class-dependencies in this model, however, is by allowing for some parameters to be *shared* between classes. In most natural problems, inputs from different classes share the same *domain*, and therefore show similar characteristics. Consider, for example, handwritten digits, which are composed of strokes, or human faces, which are composed of facial features. The features behave like "atoms" that, by themselves, are only weakly indicative of a class; it is the *composition* of these atoms that is highly class-specific[3]. Note that parameter sharing would not be possible in models like logistic regression or SVMs, which are based on linear score functions.

In order to obtain class-invariant features, we factorize the parameter tensor $W$ as follows:

$$W_{yik} = \sum_{f=1}^{F} W_{if}^{\boldsymbol{x}} W_{yf}^{\boldsymbol{y}} W_{kf}^{\boldsymbol{h}} \tag{7}$$

The model parameters are now given by three *matrices* $W^{\boldsymbol{x}}$, $W^{\boldsymbol{y}}$, $W^{\boldsymbol{h}}$, and each component $W_{yik}$ of $W$ is defined as a three-way inner product of column vectors taken from these matrices. This factorization of a three-way parameter tensor was previously used by [3] to reduce the number of parameters in an unsupervised model of images. Plugging the factorized form for the weight tensor into the definition of the probability (Eq. 4) and re-arranging terms yields

$$p(y, \boldsymbol{h}|\boldsymbol{x}) \propto \exp\Big( \sum_f \big( \sum_i x_i W_{if}^{\boldsymbol{x}} \big) \big( \sum_k h_k W_{kf}^{\boldsymbol{h}} \big) W_{yf}^{\boldsymbol{y}} \Big) \tag{8}$$

This shows that, after factorizing, we obtain a classification decision by first *projecting* the input vector $\boldsymbol{x}$ (and the vector of hidden variables $\boldsymbol{h}$) onto $F$ basis functions, or *filters*. The resulting filter responses are multiplied and combined linearly using class-specific weights $W_{yf}^{\boldsymbol{y}}$. An illustration of the model is shown in Figure 1 (b).

As before, we need to marginalize over $\boldsymbol{h}$ to obtain class-probabilities. In analogy to Eqs. 6, we obtain the final form (here written in the log-domain):

$$\log p(y|\boldsymbol{x}) = a_y - \log \sum_{y'} \exp(a_{y'}) \tag{9}$$

where

$$a_y = \sum_k \log \Big( 1 + \exp \big( \sum_f (\sum_i x_i W_{if}^{\boldsymbol{x}}) W_{kf}^{\boldsymbol{h}} W_{yf}^{\boldsymbol{y}} \big) \Big). \tag{10}$$

Note that in this model, learning of *features* (the $F$ basis functions $W_{\cdot f}^{\boldsymbol{x}}$) is tied in with learning of the classifier itself. In contrast to neural networks and deep learners ([11], [12]), the model does not try to learn a feature hierarchy. Instead, learned features are combined *multiplicatively* with hidden variables and the results added up to provide the inputs to the class-units. In terms of neural networks nomenclature, the factored model can best be thought of as a single-hidden-layer network. In general, however, the concept of "layers" is not immediately applicable in this architecture.

## 2.4 Interpretation

An illustration of the graphical model is shown in Figure 1 (non-factored model on the left, factored model on the right). Each hidden variable $h_k$ that is "*on*" contributes a slice $W_{\cdot k \cdot}$ of the parameter tensor to a blend $\sum_k h_k W_{\cdot k \cdot}$ of at most $K$ matrices. The classification decision is the sum over *all possible* instantiations of $\boldsymbol{h}$ and thus over *all possible* such blends. A single blend is simply a linear logistic classifier.

An alternative view is that each output unit $y$ accumulates evidence for or against its class by projecting the input onto $K$ basis functions (the rows of $W_y$ in Eq. 4). Each instantiation of $\boldsymbol{h}$ constitutes one way of combining a subset of basis function responses that are considered to be *consistent* into a single piece of evidence. Marginalizing over $\boldsymbol{h}$ allows us to express the fact that there can be multiple alternative sets of consistent basis function responses. This is like using an "OR" gate to combine the responses of a set of "AND" gates, or like computing a probabilistic version of a *disjunctive normal form* (DNF). As an example, consider the task of classifying a handwritten 0 that is roughly centered in the image but rotated by a random angle (see also Section 3): Each of the following combinations: (i) a vertical stroke on the left *and* a vertical stroke on the right; (ii) a horizontal stroke on the top *and* a horizontal stroke on the bottom; (iii) a diagonal stroke on the bottom left *and* a diagonal stroke on the top right, would constitute positive evidence for class 0. The model can accomodate each if necessary by making appropriate use of the hidden variables.

The factored model, where basis function responses are computed jointly for all classes and then weighted differently for each class, can be thought of as accumulating evidence accordingly in the "spatial frequency domain".

## 2.5 Discriminative gradients

Like the class-probabilities (Eq. 5) and thus the model's objective function, the derivative of the log-probability *w.r.t.* model parameters, is tractable, and scales linearly not exponentially with $K$. The derivative *w.r.t.* to a single parameter $W_{\bar{y}ik}$ of the *unfactored* form (Section 2.2) takes the form:

$$\frac{\partial \log p(y|\boldsymbol{x})}{\partial W_{\bar{y}ik}} = \big( \delta_{\bar{y}y} - p(\bar{y}|\boldsymbol{x}) \big) \sigma \big( \sum_i x_i W_{yik} h_k \big) x_i \quad \text{with} \quad \sigma(a) = \big( 1 + \exp(-a) \big)^{-1}. \tag{11}$$

To compute gradients of the factored model (Section 2.3) we use Eq. 11 and the chain rule, in conjunction with Eq. 7:

$$\frac{\partial \log p(y|\boldsymbol{x})}{\partial W_{if}^{\boldsymbol{x}}} = \sum_{\bar{y},k} \frac{\partial \log p(y|\boldsymbol{x})}{\partial W_{\bar{y}ik}} \frac{\partial W_{\bar{y}ik}}{\partial W_{if}^{\boldsymbol{x}}}. \tag{12}$$

Similarly for $W_{yf}^{\boldsymbol{y}}$ and $W_{kf}^{\boldsymbol{h}}$ (with the sums running over the remaining indices).

As with logistic regression, we can thus perform gradient based optimization of the model likelihood for training. Moreover, since we have closed form expressions, it is possible to use conjugate gradients for fast training. However, in contrast to logistic regression, the model's objective function is non-linear, so it can contain local optima. We discuss this issue in more detail in the following section. Like logistic regression, and in contrast to SVMs, the model computes probabilities and thus provides well-calibrated estimates of uncertainty in its decisions.

## 2.6 Optimization

The log-probability is non-linear and can contain local optima *w.r.t.* $W$, so some care has to be taken to obtain good local optima during training. In general we found that simply deploying a general-purpose conjugate gradient solver on random parameter initializations does not reliably yield good local optima (even though it can provide good solutions in some cases). Similar problems occur when training neural networks.

While simple gradient descent tends to yield better results, we adopt the approach discussed in [2] in most of our experiments, which consists in initializing with class-specific optimization: The set of parameters in our proposed model is the same as the ones for an ensemble of *class-specific* distributions $p(\boldsymbol{x}|y)$ (by simply adjusting the normalization in Eq. 4). More specifically, the distribution $p(\boldsymbol{x}|y)$ of inputs given labels is a factored Restricted Boltzmann machine, that can be optimized using contrastive divergence [3]. We found that performing a few iterations of class-conditional optimization as an initialization reliably yields good local optima of the model's objective function. We also experimented with alternative approaches to avoiding bad local optima, such as letting parameters grow slowly during the optimization ("annealing"), and found that class-specific pre-training yields the best results. This pre-training is reminiscent of training deep networks, which also rely on a pre-training phase. In contrast, however, here we pre-train *class-conditionally*, and initialize the whole model at once, rather than layer-by-layer. It is possible to perform a different kind of annealing by adding the class-specific and the model's actual objective function, and slowly reducing the class-specific influence using some weighting scheme. We used both the simple and the annealed optimization in some of our experiments, but did not find clear evidence that annealing leads to better local optima. We found that, given an initialization near a local optimum of the objective function, conjugate gradients *can* significantly outperform stochastic gradient descent in terms of the speed at which one can optimize both the model's own objective function and the cost on validation data.

In practice, one can add a regularization (or "weight-decay") penalty $-\lambda\|W\|^2$ to the objective function, as is common for logistic regression and other classifiers, where $\lambda$ is chosen by cross-validation.

## 3 Experiments

We applied the Gated Softmax (GSM) classifier[4] on the benchmark classification tasks described in [11]. The benchmark consists of a set of classification problems, that are difficult, because they contain many subtle, and highly complicated, dependencies of classes on inputs. It was initially introduced to evaluate the performance of deep neural networks. Some examples tasks are illustrated in Figure 3. The benchmark consists of 8 datasets, each of which contains several thousand gray-level images of size $28 \times 28$ pixels. Training set sizes vary between 1200 and 10000. The test-sets contain 50000 examples each. There are three two-class problems ("rectangles", "rectangles-images" and "convex") and five ten-class problems (which are variations of the MNIST data-set[5]).

To train the model we make use of the approach described in Section 2.6. We do not make use of any random re-starts or other additional ways to find good local optima of the objective function. For the class-specific initializations, we use a class-specific RBM with binary observables on the datasets "rectangles", "mnist-rot", "convex" and "mnist", because they contain essentially binary inputs (or a heavily-skewed histogram), and Gaussian observables on the others. For the Gaussian case, we normalize the data to mean zero and standard-deviation one (independently in each dimension). We also tried "hybrid" approaches on some data-sets where we optimize a sum of the RBM and the model objective function, and decrease the influence of the RBM as training progresses.

### 3.1 Learning task-dependent invariances

The "rectangles" task requires the classification of rectangle images into the classes *horizontal* vs. *vertical* (some examples are shown in Figure 3 (a)). Figure 2 (left) shows random sets of 50 *rows* of the matrix $W_y$ learned by the unfactored model (class *horizontal* on the top, class *vertical* on

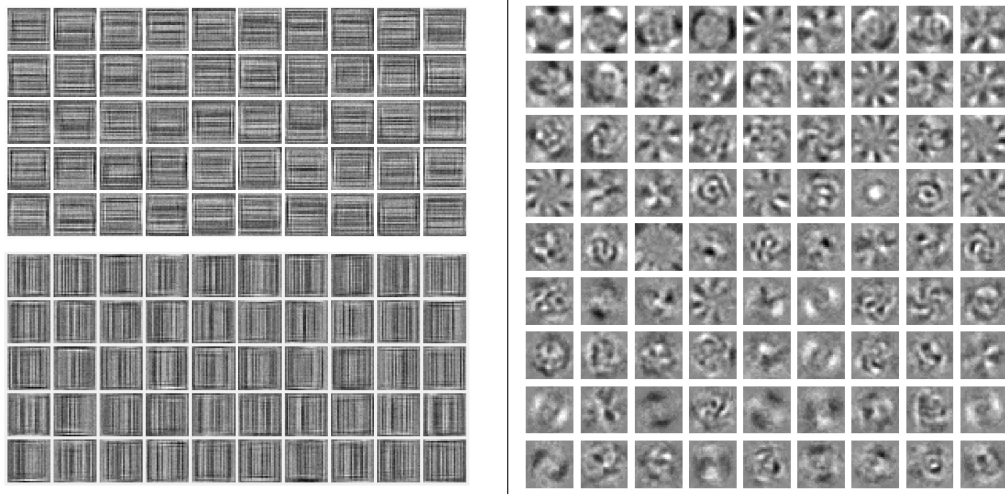

Figure 2: Left: Class-specific filters learned from the *rectangle* task – top: filters in support of the label *horizontal*, bottom: filters in support of the class label *vertical*. Right: *Shared* filters learned from *rotation-invariant digit classification*.

the bottom). Each row $W_y$ corresponds to a class-specific image filter. We display the filters using gray-levels, where brighter means larger. The plot shows that the hidden units, like "Hough-cells", make it possible to *accumulate evidence* for the different classes, by essentially counting horizontal and vertical strokes in the images. Interestingly, classification error is $0.56\%$ false, which is about a quarter the number of mis-classifications of the next best performer (SVMs with $2.15\%$ error) and significantly more accurate than all other models on this data-set.

An example of filters learned by the *factored* model is shown in Figure 2 (right). The task is classification of rotated digits in this example. Figure 3 (b) shows some example inputs. In this task, learning invariances with respect to rotation is crucial for achieving good classification performance. Interestingly, the model achieves rotation-invariance by projecting onto a set of *circular or radial Fourier*-like components. It is important to note that the model infers these filters to be the optimal input representation entirely from the task at hand. The filters resemble basis functions learned by an image transformation model trained to rotate image patches described in [3]. Classification performance is $11.75\%$ error, which is comparable with the best results on this dataset.

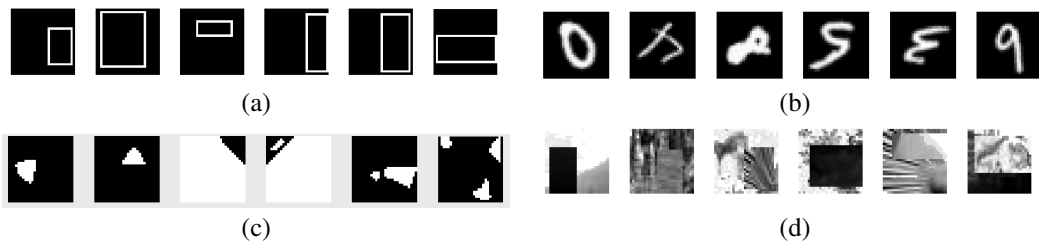

Figure 3: Example images from four of the "deep learning" benchmark tasks: (a) Rectangles (2-class): *Distinguish horizontal from vertical rectangles*; (b) Rotated digits (10-class): *Determine the class of the digit*; (c) Convex vs. non-convex (2-class): *Determine if the image shows a* convex *or* non-convex *shape*; (d) Rectangles with images (2-class): *Like (a), but rectangles are rendered using natural images*.

## 3.2 Performance

Classification performance on all $8$ datasets is shown in Figure 4. To evaluate the model we chose the number of hidden units $K$, the number of factors $F$ and the regularizer $\lambda$ based on a validation

set (typically by taking a fifth of the training set). We varied both $K$ and $F$ between 50 and 1000 on a fairly coarse grid, such as 50, 500, 1000, for most datasets, and for most cases we tried two values for the regularizer ($\lambda = 0.001$ and $\lambda = 0.0$). A finer grid may improve performance further.

Table 4 shows that the model performs well on all data-sets (comparing numbers are from [11]). It is among the best (within $0.01$ tolerance), or the best performer, in three out of 8 cases. For comparison, we also show the error rates achieved with the unfactored model (Section 2.2), which also performs fairly well as compared to deep networks and SVMs, but is significantly weaker in most cases than the factored model.

| dataset/model: | SVM | | NNet | RBM | DEEP | | GSM | |
|---|---|---|---|---|---|---|---|---|
| | SVMRBF | SVMPOL | NNet | RBM | DBN3 | SAA3 | GSM | (unfact) |
| rectangles | 2.15 | 2.15 | 7.16 | 4.71 | 2.60 | 2.41 | 0.83 | (0.56) |
| rect.-images | 24.04 | 24.05 | 33.20 | 23.69 | 22.50 | 24.05 | 22.51 | (23.17) |
| mnistplain | 3.03 | 3.69 | 4.69 | 3.94 | 3.11 | 3.46 | 3.70 | (3.98) |
| convexshapes | 19.13 | 19.82 | 32.25 | 19.92 | 18.63 | 18.41 | 17.08 | (21.03) |
| mnistbackrand | 14.58 | 16.62 | 20.04 | 9.80 | 6.73 | 11.28 | 10.48 | (11.89) |
| mnistbackimg | 22.61 | 24.01 | 27.41 | 16.15 | 16.31 | 23.00 | 23.65 | (22.07) |
| mnistrotbackimg | 55.18 | 56.41 | 62.16 | 52.21 | 47.39 | 51.93 | 55.82 | (55.16) |
| mnistrot | 11.11 | 15.42 | 18.11 | 14.69 | 10.30 | 10.30 | 11.75 | (16.15) |

Figure 4: Classification error rates on test data (error rates are in %). Models: SVMRBF: SVM with RBF kernels. SVMPOL: SVM with polynomial kernels. NNet: (MLP) Feed-forward neural net. RBM: Restricted Boltzmann Machine. DBN3: Three-layer Deep Belief Net. SAA3: Three-layer stacked auto-associator. GSM: Gated softmax model (in brackets: unfactored model).

# 4   Discussion/Future work

Several extensions of deep learning methods, including deep kernel methods, have been suggested recently (see, for example, [13], [14]), giving similar performance to the networks that we compare to here. Our method differs from these approaches in that it is not a multi-layer architecture. Instead, our model gets its power from the fact that inputs, hidden variables and labels interact in three-way cliques. Factored three-way interactions make it possible to learn task-specific features and to learn transformational invariances inherent in the task at hand.

It is interesting to note that the model outperforms kernel methods on many of these tasks. In contrast to kernel methods, the GSM provides fully probabilistic outputs and can be easily trained online, which makes it directly applicable to very large datasets.

Interestingly, the filters that the model learns (see previous Section; Figure 2) resemble those learned be recent models of image transformations (see, for example, [3]). In fact, learning of invariances in general is typically addressed in the context of learning transformations. Interestingly, most transformation models themselves are also defined via three-way interactions of some kind ([15], [16], [17], [18] , [19]). In contrast to a model of transformations, it is the *classification task* that defines the invariances here, and the model learns the invariant representations from that task only. Combining the explicit examples of transformations provided by video sequences with the implicit information about transformational invariances provided by labels is a promising future direction.

Given the probabilistic definition of the model, it would be interesting to investigate a fully Bayesian formulation that integrates over model parameters. Note that we trained the model without sparsity constraints and in a fully supervised way. Encouraging the hidden unit activities to be sparse (e.g. using the approach in [20]) and/or training the model semi-supervised are further directions for further research. Another direction is the extension to structured prediction problems, for example, by deploying the model as clique potential in a CRF.

**Acknowledgments**

We thank Peter Yianilos and the anonymous reviewers for valuable discussions and comments.

## Footnotes

[1]The *log*-probability thus decouples into a *sum* over $K$ terms and is the preferred object to compute in a numerically stable implementation.

[2]We are considering "usual" classification problems, so the number of classes is in the tens, hundreds or possibly even millions, but it is not exponential like in a CRF.

[3]If this was not the case, then many practical classification problems would be much easier to solve.

[4]An implementation of the model is available at http://*learning.cs.toronto.edu/~rfm/gatedsoftmax/*

[5]http://yann.lecun.com/exdb/mnist/

# References

[1] Julien Mairal, Francis Bach, Jean Ponce, Guillermo Sapiro, and Andrew Zisserman. Supervised dictionary learning. In *Advances in Neural Information Processing Systems 21*. 2009.

[2] Vinod Nair and Geoffrey Hinton. 3D object recognition with deep belief nets. In *Advances in Neural Information Processing Systems 22*. 2009.

[3] Roland Memisevic and Geoffrey Hinton. Learning to represent spatial transformations with factored higher-order Boltzmann machines. *Neural Computation*, 22(6):1473–92, 2010.

[4] Christopher Bishop. *Pattern Recognition and Machine Learning (Information Science and Statistics)*. Springer-Verlag New York, Inc., Secaucus, NJ, USA, 2006.

[5] Adam Berger, Vincent Della Pietra, and Stephen Della Pietra. A maximum entropy approach to natural language processing. *Computational Linguistics*, 22(1):39–71, 1996.

[6] Geoffrey Hinton. To recognize shapes, first learn to generate images. Technical report, Toronto, 2006.

[7] Hugo Larochelle and Yoshua Bengio. Classification using discriminative restricted Boltzmann machines. In *ICML '08: Proceedings of the 25th international conference on Machine learning*, New York, NY, USA, 2008. ACM.

[8] Geoffrey Hinton. Training products of experts by minimizing contrastive divergence. *Neural Computation*, 14(8):1771–1800, 2002.

[9] Roland Memisevic and Geoffrey Hinton. Unsupervised learning of image transformations. In *Proceedings of IEEE Conference on Computer Vision and Pattern Recognition*, 2007.

[10] Vinod Nair and Geoffrey Hinton. Implicit mixtures of restricted Boltzmann machines. In *Advances in Neural Information Processing Systems 21*. 2009.

[11] Hugo Larochelle, Dumitru Erhan, Aaron Courville, James Bergstra, and Yoshua Bengio. An empirical evaluation of deep architectures on problems with many factors of variation. In *ICML '07: Proceedings of the 24th international conference on Machine learning*, New York, NY, USA, 2007. ACM.

[12] Yoshua Bengio and Yann LeCun. Scaling learning algorithms towards ai. In L. Bottou, O. Chapelle, D. DeCoste, and J. Weston, editors, *Large-Scale Kernel Machines*. MIT Press, 2007.

[13] Youngmin Cho and Lawrence Saul. Kernel methods for deep learning. In *Advances in Neural Information Processing Systems 22*. 2009.

[14] Jason Weston, Frédéric Ratle, and Ronan Collobert. Deep learning via semi-supervised embedding. In *ICML '08: Proceedings of the 25th international conference on Machine learning*, New York, NY, USA, 2008. ACM.

[15] Bruno Olshausen, Charles Cadieu, Jack Culpepper, and David Warland. Bilinear models of natural images. In *SPIE Proceedings: Human Vision Electronic Imaging XII*, San Jose, 2007.

[16] Rajesh Rao and Dana Ballard. Efficient encoding of natural time varying images produces oriented space-time receptive fields. Technical report, Rochester, NY, USA, 1997.

[17] Rajesh Rao and Daniel Ruderman. Learning lie groups for invariant visual perception. In *In Advances in Neural Information Processing Systems 11*. MIT Press, 1999.

[18] David Grimes and Rajesh Rao. Bilinear sparse coding for invariant vision. *Neural Computation*, 17(1):47–73, 2005.

[19] Joshua Tenenbaum and William Freeman. Separating style and content with bilinear models. *Neural Computation*, 12(6):1247–1283, 2000.

[20] Honglak Lee, Chaitanya Ekanadham, and Andrew Ng. Sparse deep belief net model for visual area V2. In *Advances in Neural Information Processing Systems 20*. MIT Press, 2008.

